# On Invariance in Hierarchical Models

**Jake Bouvrie, Lorenzo Rosasco, and Tomaso Poggio**
Center for Biological and Computational Learning
Massachusetts Institute of Technology
Cambridge, MA USA
{jvb,lrosasco}@mit.edu, tp@ai.mit.edu

## Abstract

A goal of central importance in the study of hierarchical models for object recognition – and indeed the mammalian visual cortex – is that of understanding quantitatively the trade-off between invariance and selectivity, and how invariance and discrimination properties contribute towards providing an improved representation useful for learning from data. In this work we provide a general group-theoretic framework for characterizing and understanding invariance in a family of hierarchical models. We show that by taking an algebraic perspective, one can provide a concise set of conditions which must be met to establish invariance, as well as a constructive prescription for meeting those conditions. Analyses in specific cases of particular relevance to computer vision and text processing are given, yielding insight into how and when invariance can be achieved. We find that the minimal intrinsic properties of a hierarchical model needed to support a particular invariance can be clearly described, thereby encouraging efficient computational implementations.

## 1  Introduction

Several models of object recognition drawing inspiration from visual cortex have been developed over the past few decades [3, 8, 6, 12, 10, 9, 7], and have enjoyed substantial empirical success. A central theme found in this family of models is the use of Hubel and Wiesel's simple and complex cell ideas [5]. In the primary visual cortex, simple units compute features by looking for the occurrence of a preferred stimulus in a region of the input ("receptive field"). Translation invariance is then explicitly built into the processing pathway by way of complex units which pool locally over simple units. The alternating simple-complex filtering/pooling process is repeated, building increasingly invariant representations which are simultaneously selective for increasingly complex stimuli. In a computer implementation, the final representation can then be presented to a supervised learning algorithm.

Following the flow of processing in a hierarchy from the bottom upwards, the layerwise representations gain invariance while simultaneously becoming selective for more complex patterns. A goal of central importance in the study of such hierarchical architectures and the visual cortex alike is that of understanding quantitatively this invariance-selectivity tradeoff, and how invariance and selectivity contribute towards providing an improved representation useful for learning from examples. In this paper, we focus on hierarchical models incorporating an explicit attempt to impose transformation invariance, and do not directly address the case of deep layered models without local transformation or pooling operations (e.g. [4]).

In a recent effort, Smale et al. [11] have established a framework which makes possible a more precise characterization of the operation of hierarchical models via the study of invariance and discrimination properties. However, Smale et al. study invariance in an implicit, rather than constructive, fashion. In their work, two cases are studied: invariance with respect to image rotations and string reversals, and the analysis is tailored to the particular setting. In this paper, we reinterpret and extend the invariance analysis of Smale et al. using a group-theoretic language towards clarifying and unifying the general properties necessary for invariance in a family of hierarchical models. We show that by systematically applying algebraic tools, one can provide a concise set of conditions which must be met to establish invariance, as well as a constructive prescription for meeting those conditions. We additionally find that when one imposes the mild requirement that the transformations of interest have group structure, a broad class of hierarchical models can only be invariant to orthog-

onal transformations. This result suggests that common architectures found in the literature might need to be rethought and modified so as to allow for broader invariance possibilities. Finally, we show that our framework automatically points the way to efficient computational implementations of invariant models.

The paper is organized as follows. We first recall important definitions from Smale et al. Next, we extend the machinery of Smale et al. to a more general setting allowing for general pooling functions, and give a proof for invariance of the corresponding family of hierarchical feature maps. This contribution is key because it shows that several results in [11] do not depend on the particular choice of pooling function. We then establish a group-theoretic framework for characterizing invariance in hierarchical models expressed in terms of the objects defined here. Within this framework, we turn to the problem of invariance in two specific domains of practical relevance: images and text strings. Finally, we conclude with a few remarks summarizing the contributions and relevance of our work. *All proofs are omitted here, but can be found in the online supplementary material [2].* The reader is assumed to be familiar with introductory concepts in group theory. An excellent reference is [1].

## 2 Invariance of a Hierarchical Feature Map

We first review important definitions and concepts concerning the neural response feature map presented in Smale et al. The reader is encouraged to consult [11] for a more detailed discussion. We will draw attention to the conditions needed for the neural response to be invariant with respect to a family of arbitrary transformations, and then generalize the neural response map to allow for arbitrary pooling functions. The proof of invariance given in [11] is extended to this generalized setting. The proof presented here (and in [11]) hinges on a technical "Assumption" which must be verified to hold true, given the model and the transformations to which we would like to be invariant. Therefore the key step to establishing invariance is verification of this Assumption. After stating the Assumption and how it figures into the overall picture, we explore its verification in Section 3. There we are able to describe, for a broad range of hierarchical models (including a class of convolutional neural networks [6]), the necessary conditions for invariance to a set of transformations.

### 2.1 Definition of the Feature Map and Invariance

First consider a system of patches of increasing size associated to successive layers of the hierarchy, $v_1 \subset v_2 \subset \cdots \subset v_n \subseteq S$, with $v_n$ taken to be the size of the full input. Here layer $n$ is the top-most layer, and the patches are pieces of the *domain* on which the input data are defined. The set $S$ could contain, for example, points in $\mathbb{R}^2$ (in the case of 2D graphics) or integer indices (the case of strings). Until Section 4, the data are seen as general functions, however it is intuitively helpful to think of the special case of images, and we will use a notation that is suggestive of this particular case. Next, we'll need spaces of functions on the patches, $\text{Im}(v_i)$. In many cases it will only be necessary to work with arbitrary successive pairs of patches (layers), in which case we will denote by $u$ the smaller patch, and $v$ the next larger patch. We next introduce the transformation sets $H_i, i = 1, \dots, n$ intrinsic to the model. These are abstract sets in general, however here we will take them to be comprised of translations with $h \in H_i$ defined by $h : v_i \rightarrow v_{i+1}$. Note that by construction, the functions $h \in H_i$ implicitly involve restriction. For example, if $f \in \text{Im}(v_2)$ is an image of size $v_2$ and $h \in H_1$, then $f \circ h$ is a piece of the image of size $v_1$. The particular piece is determined by $h$. Finally, to each layer we also associate a dictionary of templates, $Q_i \subseteq \text{Im}(v_i)$. The templates could be randomly sampled from $\text{Im}(v_i)$, for example.

Given the ingredients above, the neural response $N_m(f)$ and associated derived kernel $\widehat{K}_m$ are defined as follows.

**Definition 1** (Neural Response). *Given a non-negative valued, normalized, initial reproducing kernel $\widehat{K}_1$, the $m$-th derived kernel $\widehat{K}_m$, for $m = 2, \dots, n$, is obtained by normalizing $K_m(f, g) = \langle N_m(f), N_m(g) \rangle_{L^2(Q_{m-1})}$ where $N_m(f)(q) = \max_{h \in H} \widehat{K}_{m-1}(f \circ h, q), \quad q \in Q_{m-1}$ with $H = H_{m-1}$.*

Here a kernel is normalized by taking $\widehat{K}(f, g) = K(f, g)/\sqrt{K(f, f)K(g, g)}$. Note that the neural response decomposes the input into a hierarchy of parts, analyzing sub-regions at different scales. The neural response and derived kernels describe in compact, abstract terms the core operations built into the many related hierarchical models of object recognition cited above.

We next define a set of transformations, distinct from the $H_i$ above, to which we would like to be invariant. Let $r \in \mathcal{R}_i, i \in \{1, \dots, n-1\}$, be transformations that can be viewed as mapping either $v_i$ to itself or $v_{i+1}$ to itself (depending on the context in which it is applied). We rule out the degenerate translations and transformations, $h$ or $r$ mapping their entire domain to a single point. When it is necessary to identify transformations defined on a specific domain $v$, we will use the notation $r_v : v \rightarrow v$. Invariance of the neural response feature map can now be defined.

**Definition 2** (Invariance). *The feature map $N_m$ is invariant to the domain transformation $r \in \mathcal{R}$ if $N_m(f) = N_m(f \circ r)$, for all $f \in \operatorname{Im}(v_m)$, or equivalently, $\widehat{K}_m(f \circ r, f) = 1$, for all $f \in \operatorname{Im}(v_m)$.*
In order to state the invariance properties of a given feature map, a technical assumption is needed.

**Assumption 1** (from [11]). *Fix any $r \in \mathcal{R}$. There exists a surjective map $\pi : H \to H$ satisfying*

$$r_v \circ h = \pi(h) \circ r_u \tag{1}$$

*for all $h \in H$.*

This technical assumption is best described by way of an example. Consider images and rotations: the assumption stipulates that rotating an image and then taking a restriction must be equivalent to first taking a (different) restriction and then rotating the resulting image patch. As we will describe below, establishing invariance will boil down to verifying Assumption 1.

## 2.2 Invariance and Generalized Pooling

We next provide a generalized proof of invariance of a family of hierarchical feature maps, where the properties we derive do not depend on the choice of the pooling function. Given the above assumption, invariance can be established for general pooling functions of which the $\max$ is only one particular choice. We will first define such general pooling functions, and then describe the corresponding generalized feature maps. The final step will then be to state an invariance result for the generalized feature map, given that Assumption 1 holds.

Let $H = H_i$, with $i \in \{1, \dots, n-1\}$, and let $\mathcal{B}(\mathbb{R})$ denote the Borel algebra of $\mathbb{R}$. As in Assumption 1, we define $\pi : H \to H$ to be a surjection, and let $\Psi : \mathcal{B}(\mathbb{R}_{++}) \to \mathbb{R}_{++}$ be a bounded *pooling function* defined for Borel sets $B \in \mathcal{B}(\mathbb{R})$ consisting of only positive elements. Here $\mathbb{R}_{++}$ denotes the set of strictly positive reals. Given a positive functional $F$ acting on elements of $H$, we define the set $F(H) \in \mathcal{B}(\mathbb{R})$ as

$$F(H) = \{F[h] \mid h \in H\}.$$

Note that since $\pi$ is surjective, $\pi(H) = H$, and therefore $(F \circ \pi)(H) = F(H)$.

With these definitions in hand, we can define a more general neural response as follows. For $H = H_{m-1}$ and all $q \in Q = Q_{m-1}$, let the neural response be given by

$$N_m(f)(q) = (\Psi \circ F)(H)$$

where

$$F[h] = \widehat{K}_{m-1}(f \circ h, q).$$

Given Assumption 1, we can now prove invariance of a neural response feature map built from the general pooling function $\Psi$.

**Theorem 1.** *Given any function $\Psi : \mathcal{B}(\mathbb{R}_{++}) \to \mathbb{R}_{++}$, if the initial kernel satisfies $\widehat{K}_1(f, f \circ r) = 1$ for all $r \in \mathcal{R}$, $f \in \operatorname{Im}(v_1)$, then*

$$N_m(f) = N_m(f \circ r),$$

*for all $r \in \mathcal{R}$, $f \in \operatorname{Im}(v_m)$ and $m \leq n$.*

We give a few practical examples of the pooling function $\Psi$.
**Maximum:** The original neural response is recovered setting $\Psi(B) = \sup B$.
**Averaging:** We can consider average pooling by setting $\Psi(B) = \int_{x \in B} x d\mu$. If $H$ has a measure $\rho_H$, then a natural choice for $\mu$ is the induced push-forward measure $\rho_H \circ F^{-1}$. The measure $\rho_H$ may be simply uniform, or in the case of a finite set $H$, discrete. Similarly, we may consider more general weighted averages.

## 3 A Group-Theoretic Invariance Framework

This section establishes general definitions and conditions needed to formalize a group-theoretic concept of invariance. When Assumption 1 holds, then the neural response map can be made invariant to the given set of transformations. Proving invariance thus reduces to verifying that the Assumption actually holds, and is valid. A primary goal of this paper is to place this task within an algebraic framework so that the question of verifying the Assumption can be formalized and explored in full generality with respect to model architecture, and the possible transformations. Formalization of Assumption 1 culminates in Definition 3 below, where purely algebraic conditions are separated from conditions stemming from the mechanics of the hierarchy. This separation results in a simplified problem because one can then tackle the algebraic questions independent of and untangled from the model architecture.

Our general approach is as follows. We will require that $\mathcal{R}$ is a subset of a group and then use algebraic tools to understand when and how Assumption 1 can be satisfied given different instances

of $\mathcal{R}$. If $\mathcal{R}$ is fixed, then the assumption can only be satisfied by placing requirements on the sets of built-in translations $H_i, i = 1, \ldots, n$. Therefore, we will make quantitative, constructive statements about the minimal sets of translations associated to a layer required to support invariance to a set of transformations. Conversely, one can fix $H_i$ and then ask whether the resulting feature map will be invariant to any transformations. We explore this perspective as well, particularly in the examples of Section 4, where specific problem domains are considered.

## 3.1 Formulating Conditions for Invariance

Recall that $v_i \subset S$. Because it will be necessary to translate in $S$, it is assumed that an appropriate notion of addition between the elements of $S$ is given. If $G$ is a group, we denote the (left) action of $G$ on $S$ by $A : G \times S \rightarrow S$. Given an element $g \in G$, the notation $A_g : S \rightarrow S$ will be utilized. Since $A$ is a group action, it satisfies $(A_g \circ A_{g'})(x) = A_{gg'}(x)$ for all $x \in S$ and all $g, g' \in G$. Consider an arbitrary pair of successive layers with associated patch sizes $u$ and $v$, with $u \subset v \subset S$. Recall that the definition of the neural response involves the "built-in" translation functions $h : u \rightarrow v$, for $h \in H = H_u$. Since $S$ has an addition operation, we may parameterize $h \in H$ explicitly as $h_a(x) = x + a$ for $x \in u$ and parameter $a \in v$ such that $(u + a) \subset v$. The restriction behavior of the translations in $H$ prevents us from simply generating a group out of the elements of $H$. To get around this difficulty, we will decompose the $h \in H$ into a composition of two functions: a translation group action and an inclusion.

Let $S$ generate a group of translations $T$ by defining the injective map

$$
\begin{aligned}
S &\rightarrow T \\
a &\mapsto t_a.
\end{aligned}
\tag{2}
$$

That is, to every element of $a \in S$ we associate a member of the group $T$ whose action corresponds to translation in $S$ by $a$: $A_{t_a}(x) = x + a$ for $x, a \in S$. (Although we assume the specific case of translations throughout, the sets of intrinsic operations $H_i$ may more generally contain other kinds of transformations. *We assume, however, that $T$ is abelian.*) Furthermore, because the translations $H$ can be parameterized by an element of $S$, one can apply Equation (2) to define an injective map $\tau : H \rightarrow T$ by $h_a \mapsto t_a$. Finally, we define $\iota_u : u \hookrightarrow S$ to be the canonical inclusion of $u$ into $S$. We can now rewrite $h_a : u \rightarrow v$ as

$$
h_a = A_{t_a} \circ \iota_u
$$

Note that because $a$ satisfies $(u + a) \subset v$ by definition, $\mathrm{im}(A_{t_a} \circ \iota_u) \subset v$ automatically.

In the statement of Assumption 1, the transformations $r \in \mathcal{R}$ can be seen as maps from $u$ to itself, or from $v$ to itself, depending on which side of Equation (1) they are applied. To avoid confusion we denoted the former case by $r_u$ and the latter by $r_v$. Although $r_u$ and $r_v$ are the same "kind" of transformation, one cannot in general associate to each "kind" of transformation $r \in \mathcal{R}$ a single element of some group as we did in the case of translations above. The group action could very well be different depending on the context. We will therefore consider $r_u$ and $r_v$ to be distinct transformations, loosely associated to $r$. In our development, we will make the important assumption that the transformations $r_u, r_v \in \mathcal{R}$ can be expressed as actions of elements of some group, and denote this group by $R$. More precisely, for every $r_u \in \mathcal{R}$, there is assumed to be a corresponding element $\rho_u \in R$ whose action satisfies $A_{\rho_u}(x) = r_u(x)$ for all $x \in u$, and similarly, for every $r_v \in \mathcal{R}$, there is assumed to be a corresponding element $\rho_v \in R$ whose action satisfies $A_{\rho_v}(x) = r_v(x)$ for all $x \in v$. The distinction between $\rho_u$ and $\rho_v$ will become clear in the case of feature maps defined on functions whose domain is a finite set (such as strings). In the case of images, we will see that $\rho_u = \rho_v$.

Assumption 1 requires that $r_v \circ h = h' \circ r_u$ for $h, h' \in H$, with the map $\pi : h \mapsto h'$ onto. We now restate this condition in group-theoretic terms. Define $\tilde{T} = \tau(H_u) \subseteq T$ to be the set of group elements corresponding to $H_u$. Set $h = h_a, h' = h_b$, and denote also by $r_u, r_v$ the elements of the group $R$ corresponding to the given transformation $r \in \mathcal{R}$. The Assumption says in part that $r_v \circ h = h' \circ r_u$ for some $h' \in H$. This can now be expressed as

$$
A_{r_v} \circ A_{t_a} \circ \iota_u = A_{t_b} \circ \iota_u \circ A_{r_u} \circ \iota_u
\tag{3}
$$

for some $t_b \in \tilde{T}$. In order to arrive at a purely algebraic condition for invariance, we will need to understand and manipulate compositions of group actions. However on the right-hand side of Equation (3) the translation $A_{t_b}$ is separated from the transformation $A_{r_u}$ by the inclusion $\iota_u$. We will therefore need to introduce an additional constraint on $R$. This constraint leads to our first condition for invariance: If $x \in u$, then we require that $A_{r_u}(x) \in u$ for all $r \in R$. One can now see that if this condition is met, then verifying Equation (3) reduces to checking that

$$
A_{r_v} \circ A_{t_a} = A_{t_b} \circ A_{r_u},
\tag{4}
$$

and that the map $t_a \mapsto t_b$ is onto.

The next step is to turn compositions of actions $A_x \circ A_y$ into an equivalent action of the form $A_{xy}$. Do do this, one needs $R$ and $T$ to be subgroups of the same group $G$ so that the associativity property of group actions applies. A general way to accomplish this is to form the semidirect product

$$G = T \rtimes R. \tag{5}$$

Recall that the semidirect product $G = X \rtimes Y$ is a way to put two subgroups $X, Y$ together where $X$ is required to be normal in $G$, and $X \cap Y = \{1\}$ (the usual direct product requires both subgroups to be normal). In our setting $G$ is easily shown to be isomorphic to a group with normal subgroup $T$ and subgroup $R$ where each element may be written in the form $g = tr$ for $t \in T, r \in R$. We will see below that we do not loose generality by requiring $T$ to be normal. Note that although this construction precludes $R$ from containing the transformations in $T$, allowing $R$ to contain translations is an uninteresting case.

Consider now the action $A_g$ for $g \in G = T \rtimes R$. Returning to Equation (4), we can apply the associativity property of actions and see that Equation (4) will hold as long as

$$r_v \tilde{T} = \tilde{T} r_u \tag{6}$$

for every $r \in R$. This is our second condition for invariance, and is a purely algebraic requirement concerning the groups $R$ and $T$, distinct from the restriction related conditions involving the patches $u$ and $v$.

The two invariance conditions we have described thus far combine to capture the content of Assumption 1, but in a manner that separates group related conditions from constraints due to restriction and the nested nature of an architecture's patch domains. We can summarize the invariance conditions in the form of a concise definition that can be applied to establish invariance of the neural response feature maps $N_m(f)$, $2 \le m \le n$ with respect to a set of transformations. Let $\tilde{R} \subseteq R$ be the set of transformations for which we would like to prove invariance, in correspondence with $\mathcal{R}$.

**Definition 3** (Compatible Sets). *The subsets $\tilde{R} \subset R$ and $\tilde{T} \subset T$ are* compatible *if all of the following conditions hold:*

1. *For each $r \in \tilde{R}$, $r_v \tilde{T} = \tilde{T} r_u$. When $r_u = r_v$ for all $r \in R$, this means that normalizer of $\tilde{T}$ in $\tilde{R}$ is $\tilde{R}$.*

2. *Left transformations $r_v$ never take a point in $v$ outside of $v$, and right transformations $r_u$ never take a point in $u/v$ outside of $u/v$ (respectively):*

$$im A_{r_v} \circ \iota_v \subseteq v, \qquad im A_{r_u} \circ \iota_u \subseteq u, \qquad im A_{r_u} \circ \iota_v \subseteq v,$$

*for all $r \in \tilde{R}$.*

3. *Translations never take a point in $u$ outside of $v$:*

$$im A_t \circ \iota_u \subseteq v$$

*for all $t \in \tilde{T}$.*

The final condition above has been added to ensure that any set of translations $\tilde{T}$ we might construct satisfy the implicit assumption that the hierarchy's translation functions $h \in H$ are maps which respect the definition $h : u \to v$.

If $\tilde{R}$ and $\tilde{T}$ are compatible, then for each $t_a \in \tilde{T}$ Equation 3 holds for some $t_b \in \tilde{T}$, and the map $t_a \mapsto t_b$ is surjective from $\tilde{T} \to \tilde{T}$ (by Condition (1) above). So Assumption 1 holds.

As will become clear in the following section, the tools available to us from group theory will provide insight into the structure of compatible sets.

## 3.2 Orbits and Compatible Sets

Suppose we assume that $\tilde{R}$ is a subgroup (rather than just a subset), and ask for the smallest compatible $\tilde{T}$. We will show that the only way to satisfy Condition (1) in Definition 3 is to require that $\tilde{T}$ be a union of $\tilde{R}$-*orbits*, under the action

$$(t, r) \mapsto r_v t r_u^{-1} \tag{7}$$

for $t \in T, r \in \tilde{R}$. This perspective is particularly illuminating because it will eventually allow us to view conjugation by a transformation $r$ as a permutation of $\tilde{T}$, thereby establishing surjectivity of

the map $\pi$ defined in Assumption 1. For computational reasons, viewing $\tilde{T}$ as a union of orbits is also convenient.

If $r_v = r_u = r$, then the action (7) is exactly conjugation and the $\tilde{R}$-orbit of a translation $t \in T$ is the conjugacy class $C_{\tilde{R}}(t) = \{rtr^{-1} \mid r \in \tilde{R}\}$. Orbits of this form are also equivalence classes under the relation $s \sim s'$ if $s' \in C_{\tilde{R}}(s)$, and we will require $\tilde{T}$ to be partitioned by the conjugacy classes induced by $\tilde{R}$.

The following Proposition shows that, given set of candidate translations in $H$, we can construct a set of translations compatible with $\tilde{R}$ by requiring $\tilde{T}$ to be a union of $\tilde{R}$-orbits under the action of conjugation.

**Proposition 1.** *Let $\Gamma \subseteq T$ be a given set of translations, and assume the following: (1) $G \cong T \rtimes R$, (2) For each $r \in R$, $r = r_u = r_v$, (3) $\tilde{R}$ is a subgroup of $R$. Then Condition (1) of Definition 3 is satisfied if and only if $\tilde{T}$ can be expressed as a union of orbits of the form*

$$\tilde{T} = \bigcup_{t \in \Gamma} C_{\tilde{R}}(t) \,. \tag{8}$$

An interpretation of the above Proposition, is that when $\tilde{T}$ is a union of $\tilde{R}$-orbits, conjugation by $r$ can be seen as a permutation of $\tilde{T}$. In general, a given $\tilde{T}$ may be decomposed into several such orbits and the conjugation action of $\tilde{R}$ on $\tilde{T}$ may not necessarily be transitive.

## 4 Analysis of Specific Invariances

We continue with specific examples relevant to image processing and text analysis.

### 4.1 Isometries of the Plane

Consider the case where $G$ is the group $M$ of planar isometries, $u \subset v \subset S = \mathbb{R}^2$, and $H$ involves translations in the plane. Let $O_2$ be the group of orthogonal operators, and let $t_a \in T$ denote a translation represented by the vector $a \in \mathbb{R}^2$. In this section we assume the standard basis and work with matrix representations of $G$ when it is convenient.

We first need that $T \triangleleft M$, a property that will be useful when verifying Condition (1) of Definition 3. Indeed, from the First Isomorphism Theorem [1], the quotient space $M/T$ is isomorphic to $O_2$, giving the following commutative diagram:

$$
\begin{array}{ccc}
M & \xrightarrow{\ \pi\ } & O_2 \\
{\scriptstyle \phi}\downarrow & \nearrow_{\tilde{\pi}} & \\
M/T & &
\end{array}
$$

where the isomorphism $\tilde{\pi} : M/T \to O_2$ is given by $\tilde{\pi}(mT) = \pi(m)$ and $\phi(m) = mT$. We recall that the kernel of a group homomorphism $\pi : G \to G'$ is a normal subgroup of $G$, and that normal subgroups $N$ of $G$ are invariant under the operation of conjugation by elements $g$ of $G$. That is, $gNg^{-1} = N$ for all $g \in G$. With this picture in mind, the following Lemma establishes that $T \triangleleft M$, and further shows that $M$ is isomorphic to $T \rtimes R$ with $R = O_2$, and $T$ a normal subgroup of $M$.

**Lemma 1.** *For each $m \in M$, $t_a \in T$, $mt_a = t_b m$ for some unique element $t_b \in T$.*

We are now in a position to verify the Conditions of Definition 3 for the case of planar isometries.

**Proposition 2.** *Let $H$ be the set of translations associated to an arbitrary layer of the hierarchical feature map and define the injective map $\tau : H \to T$ by $h_a \mapsto t_a$, where $a$ is a parameter characterizing the translation. Set $\Gamma = \{\tau(h) \mid h \in H\}$. Take $G = M \cong T \rtimes O_2$ as above. The sets*

$$\tilde{R} = O_2, \qquad \tilde{T} = \bigcup_{t \in \Gamma} C_{\tilde{R}}(t)$$

*are compatible.*

This proposition states that the hierarchical feature map may be made invariant to isometries, however one might reasonably ask whether the feature map can be invariant to other transformations. The following Proposition confirms that isometries are the *only* possible transformations, with group structure, to which the hierarchy may be made invariant in the exact sense of Definition 2.

**Proposition 3.** *Assume that the input spaces $\{\mathrm{Im}(v_i)\}_{i=1}^{n-1}$ are endowed with a norm inherited from $\mathrm{Im}(v_n)$ by restriction. Then at all layers, the group of orthogonal operators $O_2$ is the only group of transformations to which the neural response can be invariant.*

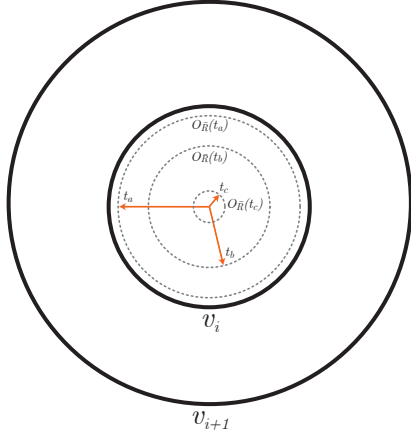

Figure 1: Example illustrating construction of an appropriate $H$. Suppose $H$ initially contains the translations $\Gamma = \{h_a, h_b, h_c\}$. Then to be invariant to rotations, the condition on $H$ is that $H$ must also include translations defined by the $\tilde{R}$-orbits $O_{\tilde{R}}(t_a), O_{\tilde{R}}(t_b)$ and $O_{\tilde{R}}(t_c)$. In this example $\tilde{R} = SO_2$, and the orbits are translations to points lying on a circle in the plane.

The following Corollary is immediate:

**Corollary 1.** *The neural response cannot be scale invariant, even if $\widehat{K}_1$ is.*

We give a few examples illustrating the application of the Propositions above.

**Example 1.** *If we choose the group of rotations of the plane by setting $\tilde{R} = SO_2 \lhd O_2$, then the orbits $O_{\tilde{R}}(a)$ are circles of radius $\|a\|$. See Figure 1. Therefore rotation invariance is possible as long as the set $\tilde{T}$ (and therefore $H$, since we can take $H = \tau^{-1}(\tilde{T})$) includes translations to all points along the circle of radius $a$, for each element $t_a \in \tilde{T}$. In particular if $H$ includes all possible translations, then Assumption 1 is verified, and we can apply Theorem 1: $N_m$ will be invariant to rotations as long as $\widehat{K}_1$ is. A similar argument can be made for reflection invariance, as any rotation can be built out of the composition of two reflections.*

**Example 2.** *Analogous to the previous example, we may also consider finite cyclical groups $C_n$ describing rotations by $\theta = 2\pi/n$. In this case the construction of an appropriate set of translations is similar: we require that $\tilde{T}$ include at least the conjugacy classes with respect to the group $C_n$, $C_{C_n}(t)$ for each $t \in \Gamma = \tau(H)$.*

**Example 3.** *Consider a simple* convolutional neural network *[6] consisting of two layers, one filter at the first convolution layer, and downsampling at the second layer defined by summation over all distinct $k \times k$ blocks. In this case, Proposition 2 and Theorem 1 together say that if the filter kernel is rotation invariant, then the output representation will be invariant to global rotation of the input image. This is so because convolution implies the choice $K_1(f, g) = \langle f, g \rangle_{L_2}$, average pooling, and $H = H_1$ containing all possible translations. If the convolution filter $z$ is rotation invariant, $z \circ r = z$ for all rotations $r$, and $K_1(f \circ r, z) = K_1(f, z \circ r^{-1}) = K_1(f, z)$. So we can conclude invariance of the initial kernel.*

### 4.2 Strings, Reflections, and Finite Groups

We next consider the case of finite length strings defined on a finite alphabet. One of the advantages group theory provides in the case of string data is that we need not work with permutation representations. Indeed, we may equivalently work with group elements which act on strings as abstract objects. The definition of the neural response given in Smale et al. involves translating an analysis window over the length of a given string. Clearly translations over a finite string do not constitute a group as the law of composition is not closed in this case. We will get around this difficulty by first considering *closed words* formed by joining the free ends of a string. Following the case of circular data where arbitrary translations are allowed, we will then consider the original setting described in Smale et al. in which strings are finite non-circular objects.

Taking a *geometric* standpoint sheds light on groups of transformations applicable to strings. In particular, one can interpret the operation of the translations in $H$ as a circular shift of a string followed by truncation outside of a fixed window. The cyclic group of circular shifts of an $n$-string is readily seen to be isomorphic to the group of rotations of an $n$-sided regular polygon. Similarly, reversal of an $n$-string is isomorphic to reflection of an $n$-sided polygon, and describes a cyclic group of order two. As in Equation (5), we can combine rotation and reflection via a semidirect product

$$D_n \cong C_n \rtimes C_2 \qquad (9)$$

where $C_k$ denotes the cyclic group of order $k$. The resulting product group has a familiar presentation. Let $t, r$ be the generators of the group, with $r$ corresponding to reflection (reversal), and $t$ corresponding to a rotation by angle $2\pi/n$ (leftward circular shift by one character). Then the group of symmetries of a closed $n$-string is described by the relations

$$D_n = \langle t, r \mid t^n, r_v^2, r_v t r_v t \rangle. \tag{10}$$

These relations can be seen as describing the ways in which an $n$-string can be left unchanged. The first says that circularly shifting an $n$-string $n$ times gives us back the original string. The second says that reflecting twice gives back the original string, and the third says that left-shifting then reflecting is the same as reflecting and then right-shifting. In describing exhaustively the symmetries of an $n$-string, we have described exactly the dihedral group $D_n$ of symmetries of an $n$-sided regular polygon. As manipulations of a closed $n$-string and an $n$-sided polygon are isomorphic, we will use geometric concepts and terminology to establish invariance of the neural response defined on strings with respect to reversal. In the following discussion we will abuse notation and at times denote by $u$ and $v$ the largest index associated with the patches $u$ and $v$.

In the case of reflections of strings, $r_u$ is quite distinct from $r_v$. The latter reflection, $r_v$, is the usual reflection of an $v$-sided regular polygon, whereas we would like $r_u$ to reflect a smaller $u$-sided polygon. To build a group out of such operations, however, we will need to ensure that $r_u$ and $r_v$ both apply in the context of $v$-sided polygons. This can be done by extending $A_{r_u}$ to $v$ by defining $r_u$ to be the composition of two operations: one which reflects the $u$ portion of a string and leaves the rest fixed, and another which reflects the remaining $(v - u)$-substring while leaving the first $u$-substring fixed. In this case, one will notice that $r_u$ can be written in terms of rotations and the usual reflection $r_v$:

$$r_u = r_v t^{-u} = t^u r_v. \tag{11}$$

This also implies that for any $x \in T$,

$$\{rxr^{-1} \mid r \in \langle r_v \rangle\} = \{rxr^{-1} \mid r \in \langle r_v, r_u \rangle\},$$

where we have used the fact that $T$ is abelian, and applied the relations in Equation (10). We can now make an educated guess as to the form of $\tilde{T}$ by starting with Condition (1) of Definition 3 and applying the relations appearing in Equation (10). Given $x \in \tilde{T}$, a reasonable requirement is that there must exist an $x' \in \tilde{T}$ such that $r_v x = x' r_u$. In this case

$$x' = r_v x r_u = r_v x r_v t^{-u} = x^{-1} r_v r_v t^{-u} = x^{-1} t^{-u}, \tag{12}$$

where the second equality follows from Equation (11), and the remaining equalities follow from the relations (10). The following Proposition confirms that this choice of $\tilde{T}$ is compatible with the reflection subgroup of $G = D_v$, and closely parallels Proposition 2.

**Proposition 4.** *Let $H$ be the set of translations associated to an arbitrary layer of the hierarchical feature map and define the injective map $\tau : H \to T$ by $h_a \mapsto t_a$, where $a$ is a parameter characterizing the translation. Set $\Gamma = \{\tau(h) \mid h \in H\}$. Take $G = D_n \cong T \rtimes R$, with $T = C_n = \langle t \rangle$ and $R = C_2 = \{r, 1\}$. The sets*

$$\tilde{R} = R, \qquad \tilde{T} = \Gamma \cup \Gamma^{-1} t^{-u}$$

*are compatible.*

One may also consider non-closed strings, as in Smale et al., in which case substrings which would wrap around the edges are disallowed. Proposition 4 in fact points to the minimum $\tilde{T}$ for reversals in this scenario as well, noticing that the set of allowed translations is the same set above but with the illegal elements removed. If we again take length $u$ substrings of length $v$ strings, this reduced set of valid transformations in fact describes the symmetries of a regular $(v - u + 1)$-gon. We can thus apply Proposition 4 working with the Dihedral group $G = D_{v-u+1}$ to settle the case of non-closed strings.

## 5 Conclusion

We have shown that the tools offered by group theory can be profitably applied towards understanding invariance properties of a broad class of deep, hierarchical models. If one knows in advance the transformations to which a model should be invariant, then the translations which must be built into the hierarchy can be described. In the case of images, we showed that the only group to which a model in the class of interest can be invariant is the group of planar orthogonal operators.

### Acknowledgments

This research was supported by DARPA contract FA8650-06-C-7632, Sony, and King Abdullah University of Science and Technology.

# References

[1] M. Artin. *Algebra*. Prentice-Hall, 1991.

[2] J. Bouvrie, L. Rosasco, and T. Poggio. Supplementary material for "On Invariance in Hierarchical Models". *NIPS*, 2009. Available online: `http://cbcl.mit.edu/publications/ps/978_supplement.pdf`.

[3] K. Fukushima. Neocognitron: A self-organizing neural network model for a mechanism of pattern recognition unaffected by shift in position. *Biol. Cyb.*, 36:193–202, 1980.

[4] G.E. Hinton and R.R. Salakhutdinov. Reducing the dimensionality of data with neural networks. *Science*, 313(5786):504–507, 2006.

[5] D.H. Hubel and T.N. Wiesel. Receptive fields and functional architecture of monkey striate cortex. *J. Phys.*, 195:215–243, 1968.

[6] Y. LeCun, L. Bottou, Y. Bengio, and P. Haffner. Gradient-based learning applied to document recognition. *Proc. of the IEEE*, 86(11):2278–2324, November 1998.

[7] H. Lee, R. Grosse, R. Ranganath, and A. Ng. Convolutional deep belief networks for scalable unsupervised learning of hierarchical representations. In *Proceedings of the Twenty-Sixth International Conference on Machine Learning*, 2009.

[8] B.W. Mel. SEEMORE: Combining color, shape, and texture histogramming in a neurally inspired approach to visual object recognition. *Neural Comp.*, 9:777–804, 1997.

[9] T. Serre, A. Oliva, and T. Poggio. A feedforward architecture accounts for rapid categorization. *Proceedings of the National Academy of Science*, 104:6424–6429, 2007.

[10] T. Serre, L. Wolf, S. Bileschi, M. Riesenhuber, and T. Poggio. Robust object recognition with cortex-like mechanisms. *IEEE Trans. on Pattern Analysis and Machine Intelligence*, 29:411–426, 2007.

[11] S. Smale, L. Rosasco, J. Bouvrie, A. Caponnetto, and T. Poggio. Mathematics of the neural response. *Foundations of Computational Mathematics*, June 2009. available online, DOI:10.1007/s10208-009-9049-1.

[12] H. Wersing and E. Korner. Learning optimized features for hierarchical models of invariant object recognition. *Neural Comput.*, 7(15):1559–1588, July 2003.

